# Synergy and redundancy among brain cells of behaving monkeys

**Itay Gat** *
Institute of Computer Science and
Center for Neural Computation
The Hebrew University, Jerusalem 91904, Israel

**Naftali Tishby** [†]
NEC Research Institute
4 Independence Way
Princeton NJ 08540

## Abstract

Determining the relationship between the activity of a single nerve cell to that of an entire population is a fundamental question that bears on the basic neural computation paradigms. In this paper we apply an information theoretic approach to quantify the level of cooperative activity among cells in a behavioral context. It is possible to discriminate between synergetic activity of the cells vs. redundant activity, depending on the difference between the information they provide when measured jointly and the information they provide independently. We define a synergy value that is positive in the first case and negative in the second and show that the synergy value can be measured by detecting the behavioral mode of the animal from simultaneously recorded activity of the cells. We observe that among cortical cells positive synergy can be found, while cells from the basal ganglia, active during the same task, do not exhibit similar synergetic activity.

{itay,tishby}@cs.huji.ac.il
Permanent address: Institute of Computer Science and Center for Neural Computation, The Hebrew University, Jerusalem 91904, Israel.

# 1   Introduction

Measuring ways by which several neurons in the brain participate in a specific computational task can shed light on fundamental neural information processing mechanisms. While it is unlikely that complete information from any macroscopic neural tissue will ever be available, some interesting insight can be obtained from simultaneously recorded cells in the cortex of behaving animals. The question we address in this study is the level of synergy, or the level of cooperation, among brain cells, as determined by the information they provide about the observed behavior of the animal.

## 1.1   The experimental data

We analyze simultaneously recorded units from behaving monkeys during a delayed response behavioral experiment. The data was collected at the high brain function laboratory of the Haddassah Medical School of the Hebrew university[1, 2]. In this task the monkey had to remember the location of a visual stimulus and respond by touching that location after a delay of 1-32 sec. Correct responses were rewarded by a drop of juice. In one set of recordings six micro-electrodes were inserted simultaneously to the frontal or prefrontal cortex[1, 3]. In another set of experiments the same behavioral paradigm was used and recording were taken from the striatum - which is the first station in basal ganglia (a sub-cortical ganglia)[2]. The cells recorded in the striatum were the tonically active neurons[2], which are known to be the cholinergic inter-neurons of the striatum. These cells are known to respond to reward.

The monkeys were trained to perform the task in two alternating modes, "Go" and "No-Go"[1]. Both sets of behavioral modes can be detected from the recorded spike trains using several statistical modeling techniques that include Hidden Markov Models (HMM) and Post Stimulus Histograms (PSTH). The details of these detection methods are reported elsewhere[4, 5]. For this paper it is important to know that we can significantly detect the correct behavior, for example in the "Go" vs. the "No-Go" correct detection is achieved about 90% of the time, where the random is 50% and the monkey's average performance is 95% correct on this task.

# 2   Theoretical background

Our measure of synergy level among cells is information theoretic and was recently proposed by Brenner *et. al.*[6] for analysis of spikes generated by a single neuron. This is the first application of this measure to quantify cooperativity among neurons.

## 2.1   Synergy and redundancy

A fundamental quantity in information theory is the mutual information between two random variables $X$ and $Y$. It is defined as the cross-entropy (Kullbak-Liebler divergence) between the joint distribution of the variables, $p(x, y)$, and the product of the marginal distributions $p(x)p(y)$. As such it measures the statistical dependence of the variables $X$ and $Y$. It is symmetric in $X$ and $Y$ and has the following

familiar relations to their entropies[7]:

$$I(X;Y) = D_{KL}[P(X,Y)|P(X)P(Y)] = \sum_{x,y} P(x,y) \log \left( \frac{p(x,y)}{p(x)p(y)} \right) \qquad (1)$$
$$= H(X) + H(Y) - H(X,Y) = H(X) - H(X|Y) = H(Y) - H(Y|X).$$

When given three random variables $X_1$, $X_2$ and $Y$, one can consider the mutual information between the joint variables $(X_1, X_2)$ and the variable $Y$, $I(X_1, X_2; Y)$ (notice the position of the semicolon), as well as the mutual informations $I(X_1; Y)$ and $I(X_2; Y)$. Similarly, one can consider the mutual information between $X_1$ and $X_2$ *conditioned* on a given value of $Y = y$, $I(X_1; X_2|y) = D_{KL}[P(X_1, X_2|y)|P(X_1|y)P(X_2|y)]$, as well as its average, the conditional mutual information,

$$I(X_1; X_2|Y) = \sum_y p(y) I_y(X_1; X_2).$$

Following Brenner *et. al.*[6] we define the *synergy level* of $X_1$ and $X_2$ with respect to the variable $Y$ as

$$Syn_Y(X_1, X_2) = I(X_1, X_2; Y) - (I(X_1; Y) + I(X_2; Y)), \qquad (2)$$

with the natural generalization to more than two variables $X$. This expression can be rewritten in terms of entropies and conditional information as follows:

$$Syn_Y(X_1, X_2) = \qquad\qquad\qquad\qquad\qquad\qquad\qquad\qquad (3)$$
$$= H(X_1, X_2) - H(X_1, X_2|Y) - ((H(X_1) - H(X_1|Y)) + (H(X_2) - H(X_2|Y)))$$
$$= \underbrace{H(X_1|Y) + H(X_2|Y) - H(X_1, X_2|Y)}_{Depends\ On\ Y} + \underbrace{H(X_1, X_2) - (H(X_1) + H(X_2))}_{Independent\ of\ Y}$$
$$= I(X_1; X_2|Y) - I(X_1; X_2).$$

When the variables exhibit positive synergy value, with respect to the variable $Y$, they jointly provide more information on $Y$ than when considered independently, as expected in *synergetic* cases. Negative synergy values correspond to *redundancy* - the variables do not provide independent information about $Y$. Zero synergy value is obtained when the variables are independent of $Y$ or when there is no change in their dependence when conditioned on $Y$. We claim that this is a useful measure of cooperativity among neurons, in a given computational task.

It is clear from Eq.( 3) that if

$$I_y(X_1; X_2) = I(X_1; X_2) \ \forall y \in Y \Rightarrow Syn_Y(X_1, X_2) = 0, \qquad (4)$$

since in that case $\sum_y p(y) I_y(X_1; X_2) = I(X_1; X2)$.

In other words, the synergy value is not zero only if the statistical dependence, hence the mutual information between the variables, is affected by the value of $Y$. It is positive when the mutual information increase, on the average, when conditioned on $Y$, and negative if this conditional mutual information decrease. Notice that the value of synergy can be both positive and negative since information, unlike entropy, is not sub-additive in the $X$ variables.

# 3 Synergy among neurons

Our measure of synergy among the units is based on the ability to detect the behavioral mode from the recorded activity, as we discuss bellow. As discussed above, synergy among neurons is possible only if their statistical dependence change with time. An important case where synergy is not expected is pure "population coding"[8]. In this case the cells are expected to fire independently, each with its own fixed tuning curve. Our synergy value can thus be used to test if the recorded units are indeed participating in a pure population code of this kind, as hypothesized for certain motor cortical activity.

Theoretical models of the cortex that clearly predict nonzero synergy include attractor neural networks (ANN)[9] and synfire chain models(SFC)[3]. Both these models predict changes in the collective activity patterns, as neurons move between attractors in the ANN case, or when different synfire-chains of activity are born or disappear in the SFC case. To the extent that such changes in the collective activity depend on behavior, nonzero synergy values can be detected. It remains an interesting theoretical challenge to estimate the quantitative synergy values for such models and compare it to observed quantities.

## 3.1 Time-dependent cross correlations

In our previous studies[4] we demonstrated, using hidden Markov models of the activity, that the pairwise cross-correlations in the same data can change significantly with time, depending on the underlying collective state of activity. These states, revealed by the hidden Markov model, in turn depend on the behavior and enable its prediction. Dramatic and fast changes in the cross-correlation of cells has also been shown by others[10]. This finding indicate directly that the statistical dependence of the neurons can change (rapidly) with time, in a way correlated to behavior. This clearly suggests that nonzero synergy should be observed among these cortical units, relative to this behavior. In the present study this theoretical hypothesis is verified.

## 3.2 Redundancy cases

If on the other hand the conditioned mutual information equal zero for all behavioral modes, i.e. $I_y(X_1; X_2) = 0 \ \forall y \in Y$, while $I(X_1; X_2) > 0$, we expect to get negative synergy, or redundancy among the cells, with respect to the behavior variable $Y$. We observed clear redundancy in another part of the brain, the basal ganglia, during the same experiment, when the behavior was the pre-reward and post-reward activity. In this case different cells provide exactly the same information, which yields negative synergy values.

# 4 Experimental results

## 4.1 Synergy measurement in practice

To evaluate the synergy value among different cells, it is necessary to estimate the conditional distribution $p(y|x)$ where $y$ is the current behavior and $x$ represent a single trial of spike trains of the considered cells. Estimating this probability,

however, requires an underlying statistical model, or a represented of the spike trains. Otherwise there is never enough data since cortical spike trains are never exactly reproducible. In this work we choose the rate representation, which is the simplest to evaluate. The estimation of $p(y|x)$ goes as follows:

- For each of the $M$ behavioral modes $(y_1, y_2 .., y_M)$ collect spike train samples (the *training* data set).

- Using the training sample, construct a Post Stimulus Time Histogram (PSTH), i.e. the rate as function of time, for each behavioral mode.

- Given a spike train, outside of the training set, compute its probability to be result in each of the $M$ modes.

- The spike train considered correctly classified if the most probable mode is in fact the true behavioral mode, and incorrectly otherwise.

- The fraction of correct classification, for all spike trains of a given behavioral mode $y_i$, is taken as the estimate of $p(y_i|x)$, and denoted $P_{c_i}$, where $c_i$ is the identity of the cells used in the computation.

For the case of only two categories of behavior and for a uniform distribution of the different categories, the value of the entropy $H(Y)$ is the same for all combinations of cells, and is simply $H(Y) = -\sum_y p(y) \log_2(p(y)) = \log_2 2 = 1$. The full expression (in bits) for the synergy value can be thus written as follows:

$$\sum_x p(x) \left[ -\sum_y P_{c_1,c_2} \log_2(P_{c_1,c_2}) \right] \overset{?}{>} \tag{5}$$
$$1 + \sum_x p(x) \left[ -\sum_y P_{c_1} \log_2(P_{c_1}) \right] + \sum_x p(x) \left[ -\sum_y P_{c_2} \log_2(P_{c_2}) \right] ,$$

If the first expression is larger than the second than there is (positive) synergy and vice versa for redundancy. However there is one very important caveat. As we saw the computation of the mutual information is not done exactly, and what one really computes is only a lower bound. If the bound is tighter for multiple cell calculation, the method could falsely infer positive synergy, and if the bound is tighter for the single cell computation, the method could falsely infer negative synergy. In previous works we have shown that the method we use for this estimation is quite reasonable and robust[5], therefore, we believe that we have even a conservative (i.e. less positive) estimate of synergy.

## 4.2 Observed synergy values

In the first set of experiments we tried to detect the behavioral mode during the delay-period of correct trials. In this case the two types of behavior were the "Go" and the "No-Go" described in the introduction. An example of this detection problem is given in figure 1A. In this figure there are 100 examples of multi-electrode recording of spike trains during the delay period. On the left is the "Go-mode" data and on the right the "No-Go mode", for two cells. On the lower part there is an example of two single spike trains that need to be classified by the mode models.

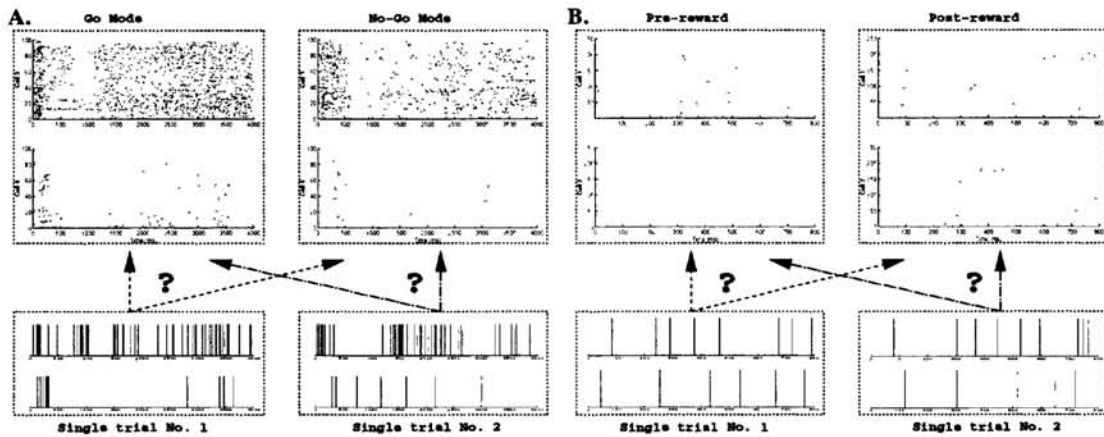

Figure 1: Raster displays of simultaneously recorded cells in the 2 different areas, in each area there were 2 behavioral modes.

Table 1 gives some examples of detection results obtained by using 2 cells independently, and by using their joint combination. It can be seen that the synergy is positive and significant. We examined 19 recording session of the same behavioral modes for two different animals and evaluated the synergy value. In 18 out of the 19 sessions there was at least one example of significant positive synergy among the cells.

For comparison we analyzed another set of experiments in which the data was recorded from the striatum in the basal ganglia. An example for this detection is shown in figure 1B. The behavioral modes were the "pre-reward" vs. the "post-reward" periods. Nine recording sessions for the two different monkeys were examined using the same detection technique. Although the detection results improve when the number of cells increase, in none of these recordings a positive synergy value was found. For most of the data the synergy value was close to zero, i.e. the mutual information among two cells jointly was close to the sum of the mutual information of the independent cells, as expected when the cells exhibit (conditionally) independent activity.

The prevailing difference between the synergy measurements in the cortex and in the TANs' of the basal ganglia is also strengthen by the different mechanisms underlying those cells. The TANs' are assumed to be globally mediators of information in the striatum, a relatively simple task, whereas the information processed in the frontal cortex in this task is believed to be much more collective and complicated. Here we suggest a first handle for quantitative detection of such different neuronal activities.

## Acknowledgments

Special thanks are due to Moshe Abeles for his encouragement and support, and to William Bialek for suggesting the idea to look for the synergy among cortical cells. We would also like to thank A. Raz, Hagai Bergman, and Eilon Vaadia for sharing their data with us. The research at the Hebrew university was supported in part by a grant from the Unites States Israeli Binational Science Foundation (BSF).

Table 1: Examples of synergy among cortical neurons. For each example the mutual information of each cell separately is given together with the mutual information of the pair. In parenthesis the matching detection probability (average over $p(y|x)$) is also given. The last column gives the percentage of increase from the mutual information of the single cells to the mutual information of the pair. The table gives only those pairs for which the percentage was larger than 20% and the detection rate higher than 60%.

| Session | Cells | Cell1 | Cell2 | Both cells | Syn (%) |
|---|---|---|---|---|---|
| bl16b | 5,6 | 0.068 (64.84) | 0.083 (66.80) | 0.209 (76.17) | 38 |
| bl21b | 1,4 | 0.201 (73.74) | 0.118 (69.70) | 0.497 (87.88) | 56 |
| bl21b | 3,4 | 0.082 (66.67) | 0.118 (69.70) | 0.240 (77.78) | 20 |
| bl26b | 0,3 | 0.062 (62.63) | 0.077 (66.16) | 0.198 (75.25) | 42 |
| bl26b | 1,2 | 0.030 (60.10) | 0.051 (63.13) | 0.148 (72.22) | 82 |
| cl77b | 2,3 | 0.054 (62.74) | 0.013 (61.50) | 0.081 (68.01) | 20 |
| cr38b | 0,2 | 0.074 (65.93) | 0.058 (63.19) | 0.160 (73.08) | 21 |
| cr38b | 0,4 | 0.074 (65.93) | 0.042 (62.09) | 0.144 (71.98) | 24 |
| cr38b | 3,4 | 0.051 (62.09) | 0.042 (62.09) | 0.111 (69.23) | 20 |
| cr43b | 0,1 | 0.070 (65.00) | 0.063 (64.44) | 0.181 (74.44) | 36 |

# References

[1] M. Abeles, E. Vaadia, H. Bergman, *Firing patterns of single unit in the prefrontal cortex and neural-networks models.*, Network 1 (1990).

[2] E. Raz, *et al Neuronal synchronization of tonically active neurons in the striatum of normal and parkinsonian primates*, J. Neurophysiol. 76:2083-2088 (1996).

[3] M. Abeles, *Corticonics*, (Cambridge University Press, 1991).

[4] I. Gat, N. Tishby and M. Abeles, *Hidden Markov modeling of simultaneously recorded cells in the associative cortex of behaving monkeys*, Network,8:297–322 (1997).

[5] I. Gat, N. Tishby, *Comparative study of different supervised detection methods of simultaneously recorded spike trains*, in preparation.

[6] N. Brenner, S.P. Strong, R. Koberle, W. Bialek, and R. de Ruyter van Steveninck, *The Economy of Impulses and the Stiffnes of Spike Trains*, NEC Research Institute Technical Note (1998).

[7] T.M. Cover and J.A. Thomas, *Elements of Information Theory.*, (Wiley NY, 1991).

[8] A.P. Georgopoulos, A.B. Schwartz, R.E. Kettner, *Neuronal Population Coding of Movement Direction*, Science, 233:1416-1419 (1986).

[9] D.J. Amit, *Modeling Brain Function*, (Cambridge University Press, 1989).

[10] E. Ahissar *et al Dependence of Cortical Plasticity on Correlated Activity of Single Neurons and on Behavioral Context*, Science, 257:1412-1415 (1992).